# Rates of Convergence of Performance Gradient Estimates Using Function Approximation and Bias in Reinforcement Learning

**Gregory Z. Grudic**
University of Colorado, Boulder
*grudic@cs.colorado.edu*

**Lyle H. Ungar**
University of Pennsylvania
*ungar@cis.upenn.edu*

## Abstract

We address two open theoretical questions in Policy Gradient Reinforcement Learning. The first concerns the efficacy of using function approximation to represent the state action value function, $Q$. Theory is presented showing that linear function approximation representations of $Q$ can *degrade* the rate of convergence of performance gradient estimates by a factor of $O(ML)$ relative to when no function approximation of $Q$ is used, where $M$ is the number of possible actions and $L$ is the number of basis functions in the function approximation representation. The second concerns the use of a bias term in estimating the state action value function. Theory is presented showing that a non-zero bias term can *improve* the rate of convergence of performance gradient estimates by $O(1 - (1/M))$, where $M$ is the number of possible actions. Experimental evidence is presented showing that these theoretical results lead to significant improvement in the convergence properties of Policy Gradient Reinforcement Learning algorithms.

## 1  Introduction

Policy Gradient Reinforcement Learning (PGRL) algorithms have recently received attention because of their potential usefulness in addressing large continuous reinforcement Learning (RL) problems. However, there is still no widespread agreement on how PGRL algorithms should be implemented. In PGRL, the agent's policy is characterized by a set of parameters which in turn implies a parameterization of the agent's performance metric. Thus if $\theta \in \Re^d$ represents a $d$ dimensional parameterization of the agent's policy and $\rho$ is a performance metric the agent is meant to maximize, then the performance metric must have the form $\rho(\theta)$ [6]. PGRL algorithms work by first estimating the performance gradient (PG) $\partial\rho/\partial\theta$ and then using this gradient to update the agent's policy using:

$$\theta_{t+1} = \theta_t + \alpha\frac{\widehat{\partial\rho}}{\partial\theta} \qquad (1)$$

where $\alpha$ is a small positive step size. If the estimate of $\partial\rho/\partial\theta$ is accurate, then the agent can climb the performance gradient in the $\theta$ parameter space, toward locally optimal policies. In practice, $\partial\rho/\partial\theta$ is estimated using samples of the state action value function $Q$. The PGRL formulation is attractive because 1) the parameterization $\theta$ of the policy can directly imply

a generalization over the agent's state space (e.g., $\theta$ can represent the adjustable weights in a neural network approximation), which suggests that PGRL algorithms can work well on very high dimensional problems [3]; 2) the computational cost of estimating $\partial\rho/\partial\theta$ is linear in the number of parameters $\theta$, which contrasts with the computational cost for most RL algorithms which grows exponentially with the dimension of the state space; and 3) PG algorithms exist which are guaranteed to give unbiased estimates of $\partial\rho/\partial\theta$ [6, 5, 4, 2, 1].

This paper addresses two open theoretical questions in PGRL formulations. In PGRL formulations performance gradient estimates typically have the following form:

$$\widehat{\frac{\partial\rho}{\partial\theta}} = f\left(\left[\hat{Q}\left(s_1, a_1\right) - b\left(s_1\right)\right], ..., \left[\hat{Q}\left(s_T, a_T\right) - b\left(s_T\right)\right]\right) \qquad (2)$$

where $\hat{Q}\left(s_i, a_i\right)$ is the estimate of the value of executing action $a_i$ in state $s_i$ (i.e. the state action value function), $b(s_i)$ the bias subtracted from $\hat{Q}\left(s_i, a_i\right)$ in state $s_i$, $T$ is the number of steps the agent takes before estimating $\partial\rho/\partial\theta$, and the form of the function $f(.)$ depends on the PGRL algorithm being used (see Section 2, equation (3) for the form being considered here). The effectiveness of PGRL algorithms strongly depends on how $\hat{Q}\left(s_i, a_i\right)$ is obtained and the form of $b(s_i)$. The aim of this work is to address these questions.

The first open theoretical question addressed here is concerned with the use of function approximation (FA) to represent the state action value function $Q$, which is in turn used to estimate the performance gradient. The original formulation of PGRL [6], the REIN-FORCE algorithm, has been largely ignored because of the slow rate of convergence of the PG estimate. The use of FA techniques to represent $Q$ based on its observations has been suggested as a way of improving convergence properties. It has been proven that specific linear FA formulations can be incorporated into PGRL algorithms, while still guaranteeing convergence to locally optimal solutions [5, 4]. However, whether linear FA representations actually improves the convergence properties of PGRL is an open question. We present theory showing that using linear basis function representations of $Q$, rather than direct observations of it, can *slow* the rate of convergence of PG estimates by a factor of $O(ML)$ (see **Theorem 1** in Section 3.1). This result suggests that PGRL formulations should avoid the use of linear FA techniques to represent $Q$. In Section 4, experimental evidence is presented supporting this conjecture.

The second open theoretical question addressed here is can a non-zero bias term $b(s)$ in (2) improve the convergence properties of PG estimates? There has been speculation that an appropriate choice of $b(s)$ can improve convergence properties [6, 5], but theoretical support has been lacking. This paper presents theory showing that if $b(s) = (1/M)\sum_a Q(s, a)$, where $M$ is the number actions, then the rate of convergence of the PG estimate is *improved* by $O(1 - (1/M))$ (see **Theorem 2** in Section 3.2). This suggests that the convergence properties of PGRL algorithms can be improved by using a bias term that is the average of $Q$ values in each state. Section 4 gives experimental evidence supporting this conjecture.

## 2   The RL Formulation and Assumptions

The RL problem is modeled as a Markov Decision Process (MDP). The agent's state at time $t \in \{1, 2, ...\}$ is given by $s_t \in S, S \subseteq \Re^D$. At each time step the agent chooses from a finite set of $M > 1$ actions $a_t \in A = a_1, ..., a_M$ and receives a reward $r_t \in \Re$. The dynamics of the environment are characterized by transition probabilities $P_{ss'}^a = Pr\{s_{t+1} = s'|s_t = s, a_t = a\}$ and expected rewards $R_s^a = E\{r_{t+1}|s_t = s, a_t = a\}, \forall s, s' \in S, a \in A$. The policy followed by the agent is characterized by a parameter vector $\theta \in \Re^d$, and is defined by the probability distribution $\pi(s, a; \theta) = Pr\{a_t = a|s_t = s; \theta\}, \forall s \in S, a \in A$. We

assume that $\pi(s, a; \theta)$ is differentiable with respect to $\theta$.

We use the Policy Gradient Theorem of Sutton et al. [5] and limit our analysis to the start state discount reward formulation. Here the reward function $\rho(\pi)$ and state action value function $Q^\pi(s, a)$ are defined as:

$$\rho(\pi) = E\left\{ \sum_{t=1}^{\infty} \gamma^t r_t \,\middle|\, s_0, \pi \right\}, \quad Q^\pi(s, a) = E\left\{ \sum_{k=1}^{\infty} \gamma^{k-1} r_{t+k} \,\middle|\, s_t = s, a_t = a, \pi \right\}$$

where $0 < \gamma \le 1$. Then the exact expression for the performance gradient is:

$$\frac{\partial \rho}{\partial \theta} = \sum_s d^\pi(s) \sum_{i=1}^{M} \frac{\partial \pi(s, a_i; \theta)}{\partial \theta} \left( Q^\pi(s, a_i) - b(s) \right) \tag{3}$$

where $d^\pi(s) = \sum_{t=0}^{\infty} \gamma^t \Pr\{s_t = s \mid s_0, \pi\}$ and $b(s) \in \Re$.

This policy gradient formulation requires that the state-action value function, $Q^\pi$, under the current policy be estimated. This estimate, $\hat{Q}^\pi$, is derived using the observed value $Q^\pi_{obs}(s, a_i)$. We assume that $Q^\pi_{obs}(s, a_i)$ has the following form:

$$Q^\pi_{obs}(s, a_i) = Q^\pi(s, a_i) + \varepsilon(s, a_i)$$

where $\varepsilon(s, a_i)$ has zero mean and finite variance $\sigma^2_{s, a_i}$. Therefore, if $\hat{Q}^\pi(s, a_i)$ is an estimate of $Q^\pi(s, a_i)$ obtained by averaging $N$ observations of $Q^\pi_{obs}(s, a_i)$, then the mean and variance are given by:

$$E\left[\hat{Q}^\pi(s, a_i)\right] = Q^\pi(s, a_i), \quad V\left[\hat{Q}^\pi(s, a_i)\right] = \frac{\sigma^2_{s, a_i}}{N} \tag{4}$$

In addition, we assume that $Q^\pi_{obs}(s, a_i)$ are independently distributed. This is consistent with the MDP assumption.

## 3  Rate of Convergence Results

Before stating the convergence theorems, we define the following:

$$\sigma^2_{\max} = \max_{s \in S, i \in \{1, \dots, M\}} \sigma^2_{s, a_i}, \quad \sigma^2_{\min} = \min_{s \in S, i \in \{1, \dots, M\}} \sigma^2_{s, a_i} \tag{5}$$

where $\sigma^2_{s, a_i}$ is defined in (4) and

$$
\begin{aligned}
C_{\min} &= \left[ \sum_s (d^\pi(s))^2 \sum_{i=1}^{M} \left( \frac{\partial \pi(s, a_i; \theta)}{\partial \theta} \right)^2 \right] \sigma^2_{\min} \\
C_{\max} &= \left[ \sum_s (d^\pi(s))^2 \sum_{i=1}^{M} \left( \frac{\partial \pi(s, a_i; \theta)}{\partial \theta} \right)^2 \right] \sigma^2_{\max}
\end{aligned}
\tag{6}
$$

### 3.1  Rate of Convergence of PIFA Algorithms

Consider the PIFA algorithm [5] which uses a basis function representation for estimated state action value function, $\hat{Q}^\pi$, of the following form:

$$\hat{Q}^\pi(s, a_i) = f^\pi_{a_i}(s) = \sum_{l=1}^{L} w_{a_i, l} \phi_{a_i, l}(s) \tag{7}$$

where $w_{a_i, l} \in \Re$ are weights and $\phi_{a_i, l}(s)$ are basis functions defined in $s \in \Re^D$. If the weights $w_{a_i, l}$ are chosen based using the observed $Q^\pi_{obs}(s, a_i)$, and the basis functions, $\phi_{a_i, l}(s)$, satisfy the conditions defined in [5, 4], then the performance gradient is given by:

$$\frac{\partial \rho}{\partial \theta}_F = \sum_s d^\pi(s) \sum_{i=1}^{M} \frac{\partial \pi(s, a_i; \theta)}{\partial \theta} f^\pi_{a_i}(s) \tag{8}$$

The following theorem establishes bounds on the rate of convergence for this representation of the performance gradient.

**Theorem 1:** *Let $\widehat{\frac{\partial \rho}{\partial \theta}}_F$ be an estimate of (8) obtained using the PIFA algorithm and the basis function representation (7). Then, given the assumptions defined in Section 2 and equations (5) and (6), the rate of convergence of a PIFA algorithm is bounded below and above by:*

$$C_{\min} \frac{ML}{N} \leq V\left[\widehat{\frac{\partial \rho}{\partial \theta}}_F\right] \leq C_{\max} \frac{ML}{N} \tag{9}$$

*where $L$ is the number of basis functions, $M$ is the number of possible actions, and $N$ is the number of independent estimates of the performance gradient.*

**Proof:** See Appendix.

### 3.2  Rate of Convergence of Direct Sampling Algorithms

In the previous section, the observed $Q_{obs}^{\pi}(s, a_i)$ are used to build a linear basis function representation of the state action value function, $Q^{\pi}(s, a_i)$, which is in turn used to estimate the performance gradient. In this section we establish rate of convergence bounds for performance gradient estimates that directly use the observed $Q_{obs}^{\pi}(s, a_i)$ without the intermediate step of building the FA representation. These bounds are established for the conditions $b(s) = (1/M)\sum_a Q(s, a)$ and $b(s) = 0$ in (3).

**Theorem 2:** *Let $\widehat{\frac{\partial \rho}{\partial \theta}}$ be a estimate of (3), be obtained using direct samples of $Q^{\pi}$. Then, if $b(s) = 0$, and given the assumptions defined in Section 2 and equations (5) and (6), the rate of convergence of $\widehat{\frac{\partial \rho}{\partial \theta}}$ is bounded by:*

$$C_{\min} \frac{1}{N} \leq V\left[\widehat{\frac{\partial \rho}{\partial \theta}}\right] \leq C_{\max} \frac{1}{N} \tag{10}$$

*where $N$ is the number of independent estimates of the performance gradient. If $b(s) \neq 0$ is defined as:*

$$b(s) = \frac{1}{M}\sum_{j=1}^{M} Q^{\pi}(s, a_j) \tag{11}$$

*then the rate of convergence of the performance gradient $\widehat{\frac{\partial \rho}{\partial \theta}}_b$ is bounded by:*

$$C_{\min} \frac{1}{N}\left(1 - \frac{1}{M}\right) \leq V\left[\widehat{\frac{\partial \rho}{\partial \theta}}_b\right] \leq C_{\max} \frac{1}{N}\left(1 - \frac{1}{M}\right) \tag{12}$$

*where $M$ is the number of possible actions.*

**Proof:** See Appendix.

Thus comparing (12) and (10) to (9) one can see that policy gradient algorithms such as PIFA which build FA representations of $Q$ converge by a factor of $O(ML)$ slower than algorithms which directly sample $Q$. Furthermore, if the bias term is as defined in (11), the bounds on the variance are further reduced by $O(1 - (1/M))$. In the next section experimental evidence is given showing that these theoretical consideration can be used to improve the convergence properties of PGRL algorithms.

## 4  Experiments

**The Simulated Environment:** The experiments simulate an agent episodically interacting in a continuous two dimensional environment. The agent starts each episode in the same state $\mathbf{s}_i$, and executes a finite number of steps following a policy to a fixed goal state $\mathbf{s}_G$. The stochastic policy is defined by a finite set of Gaussians, each associated with a specific

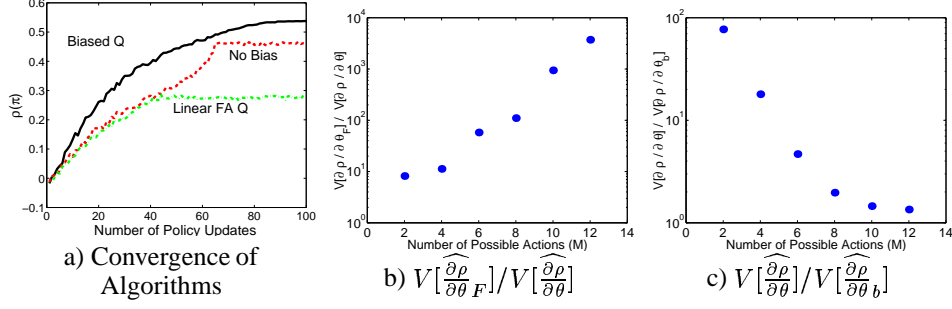

a) Convergence of Algorithms

b) $V[\widehat{\frac{\partial \rho}{\partial \theta}}_F]/V[\widehat{\frac{\partial \rho}{\partial \theta}}]$

c) $V[\widehat{\frac{\partial \rho}{\partial \theta}}]/V[\widehat{\frac{\partial \rho}{\partial \theta}}_b]$

Figure 1: Simulation Results

action. The Gaussian associated with action $a_m$ is defined as:

$$g_m\left(\mathbf{s}\right) = \exp\left[-\sum_{d=1}^{D} \frac{\left(s_d - c_{md}\right)^2}{v_{md}}\right]$$

where $\mathbf{s} = (s_1, ..., s_D) \in \Re^D$, is the agents state, $c_{m1}, ..., c_{mD}$ is the Gaussian center, and $v_{m1}, ..., v_{mD}$ is the variance along each state space dimension. The probability of executing action $a_m$ in state $\mathbf{s}$ is

$$\pi\left(\mathbf{s}, a_m; \theta\right) = \frac{g_m\left(\mathbf{s}\right)}{\sum\limits_{j=1}^{M} g_j\left(\mathbf{s}\right)}$$

where $\theta = (c_{11}, ..., c_{1d}, v_{11}, ..., v_{1d}, ..., c_{M1}, ..., c_{MD}, v_{M1}, ..., v_{MD})$ defines the policy parameters that dictate the agent's actions. Action $a_1$ directs the agent toward the goals state $\mathbf{s}_G$, while the remaining actions $a_m$ (for $m = 2, ..., M$) direct the agent towards the corresponding Gaussian center $c_{m1}, ..., c_{mD}$.

Noise is modeled using a uniform random distribution between $(0, 1)$ denoted by $U(0, 1)$, such that the noise in dimension $s_d$ is given by:

$$s_d^{obs} = s_d + \delta\left(U\left(0, 1\right) - 0.5\right)$$

where $\delta > 0$ is the magnitude of the noise, $s_d^{obs}$ is the state the agent observes and uses to choose actions, and $s_d$ is the actual state of the agent.

The agent receives a reward of $+1$ when it reaches the goal state, otherwise it receives a reward of:

$$r\left(\mathbf{s}\right) = -0.01 \exp\left[-\sum_{d=1}^{D} \frac{s_d^2}{4}\right]$$

Thus the agent gets negative rewards the closer it gets to the origin of the state space, and a positive reward whenever it reaches the goal state.

**Implementation of the PGRL algorithms:** All the PGRL formulations studied here require observations (i.e. samples) of the state action value function. $Q_{obs}^{\pi}\left(s, a_i\right)$ is sampled by executing action $a_i$ in state $s$ and thereafter following the policy. In the episodic formulation, where the agent executes a maximum of $T$ steps during each episode, at the end of each episode, $Q_{obs}^{\pi}\left(s_t, a_t\right)$ for step $t$ can be evaluated as follows:

$$Q_{obs}^{\pi}\left(s_t, a_t\right) = \sum_{k=1}^{\infty} \gamma^{k-1} r_{t+k} | s_t = s, a_t = a, \pi$$

Thus, given that the agent executes a complete episode $((s_1, a_1), ..., (s_T, a_T))$ following the policy $\pi$, at the completion of the episode we can calculate $(Q_{obs}^{\pi}(s_1, a_1), ..., Q_{obs}^{\pi}(s_T, a_T))$. This gives samples of $T$ state action value pairs. Equation (3) tells us that we require a total of $MT$ state action value function observations to estimate a performance gradient (assuming the agent can execute $M$ actions). Therefore, we can obtain the remaining $(M-1)T$ observations of $Q_{obs}^{\pi}$ by sending the agent out on

$(M-1)T$ epsiodes, each time allowing it to follow the policy $\pi$ for all $T$ steps, with the exception that action $a_t = a_m$ is executed when $Q_{obs}^\pi(s_t, a_m)$ is being observed. This sampling procedure requires a total of $(M-1)T - 1$ episodes and gives a complete set of $Q_{obs}^\pi$ state action pairs for any path $((s_1, a_1), ..., (s_T, a_T))$. For the direct sampling algorithms in Section 3.2, these observations are directly used to estimate the performance gradient. For the linear basis function based PGRL algorithm in Section 3.1, these observations are first used to calculate the $w_{a_i, l}$ as defined in [5, 4], and then the performance gradient is calculated using (8).

**Experimental Results:** Figure 1b shows a plot of average $V[\widehat{\partial\rho/\partial\theta}_F]/V[\widehat{\partial\rho/\partial\theta}]$ values over 10,000 estimates of the performance gradient. For each estimate, the goal state, start state, and Gaussian centers are all chosen using a uniform random distribution $(-1, 1)$; the Gaussian variances are sampled from a uniform distribution $(0.1, 1)$. As predicted by **Theorem 1** in Section 3.1 and **Theorem 2** in Section 3.2, as the number of actions $M$ increases, this ratio also increases. Note that Figure 1b plots average variance ratios, not the bounds in variance given in **Theorem 1** and **Theorem 2** (which have not been experimentally sampled), so the $ML$ ratio predicted by the theorems is supported by the increase in the ratio as $M$ increases. Figure 1c shows a plot of average $V[\widehat{\partial\rho/\partial\theta}]/V[\widehat{\partial\rho/\partial\theta}_b]$ values over 10,000 estimates of the performance gradient. As above, for each estimate, the goal state, start state, and Gaussian centers are all chosen using a uniform random distribution $(-1, 1)$; the Gaussian variances are sampled from a uniform distribution $(0.1, 1)$. This also follows the predicted trends of **Theorem 1** and **Theorem 2**. Finally, Figure 1a shows the average reward over 100 runs as the three algorithms converge on a two action problem. Each algorithm is given the same number of $Q_{obs}^\pi$ samples to estimate the gradient before each update. Because $\widehat{\partial\rho/\partial\theta}_b$ has the least variance, it allows the policy $\pi$ to converge to the highest reward value $\rho(\pi)$. Similarly, because $\widehat{\partial\rho/\partial\theta}_F$ has the highest variance, its policy updates converge to the worst $\rho(\pi)$. Note that because all three algorithms will converge to the same locally optimal policy given enough samples of $Q_{obs}^\pi$, Figure 1a simply demonstrates that $\widehat{\partial\rho/\partial\theta}_F$ requires more samples than $\widehat{\partial\rho/\partial\theta}$, which in turn requires more samples than $\widehat{\partial\rho/\partial\theta}_b$.

## 5  Conclusion

The theoretical and experimental results presented here indicate that how PGRL algorithms are implemented can substantially affect the number of observations of the state action value function ($Q$) needed to obtain good estimates of the performance gradient. Furthermore, they suggest that an appropriately chosen bias term, specifically the average value of $Q$ over all actions, and the direct use of observed $Q$ values can improve the convergence of PGRL algorithms. In practice linear basis function representations of $Q$ can significantly degrade the convergence properties of policy gradient algorithms. This leaves open the question of whether any (i.e. nonlinear) function approximation representation of value functions can be used to improve convergence of such algorithms.

## References

[1] Jonathan Baxter and Peter L. Bartlett, *Reinforcement learning in pomdp's via direct gradient ascent*, Proceedings of the Seventeenth International Conference on Machine Learning (ICML'2000) (Stanford University, CA), June 2000, pp. 41–48.

[2] G. Z. Grudic and L. H. Ungar, *Localizing policy gradient estimates to action transitions*, Proceedings of the Seventeenth International Conference on Machine Learning, vol. 17, Morgan Kaufmann, June 29 - July 2 2000, pp. 343–350.

[3] ______, *Localizing search in reinforcement learning*, Proceedings of the Seventeenth National Conference on Artificial Intelligence, vol. 17, Menlo Park, CA: AAAI Press / Cambridge, MA: MIT Press, July 30 - August 3 2000, pp. 590–595.

[4] V. R. Konda and J. N. Tsitsiklis, *Actor-critic algorithms*, Advances in Neural Information Processing Systems (Cambridge, MA) (S. A. Solla, T. K. Leen, and K.-R. Mller, eds.), vol. 12, MIT Press, 2000.

[5] R. S. Sutton, D. McAllester, S. Singh, and Y. Mansour, *Policy gradient methods for reinforcement learning with function approximation*, Advances in Neural Information Processing Systems (Cambridge, MA) (S. A. Solla, T. K. Leen, and K.-R. Mller, eds.), vol. 12, MIT Press, 2000.

[6] R. J. Williams, *Simple statistical gradient-following algorithms for connectionist reinforcement learning*, Machine Learning **8** (1992), no. 3, 229–256.

**Appendix: Proofs of Theorems 1 and 2**

**Proof of Theorem 1:** Consider the definition of $f_{a_i}^\pi$ given in (7). In [5] it is shown that there exist $w_{a_i,l}$ and $\phi_{a_i,l}(s)$ such that:

$$E\left[\sum_s d^\pi(s)\sum_{i=1}^M \frac{\partial\pi(s,a_i;\theta)}{\partial\theta}f_{a_i}^\pi(s) - \sum_s d^\pi(s)\sum_{i=1}^M \frac{\partial\pi(s,a_i;\theta)}{\partial\theta}Q^\pi(s,a_i)\right] = 0$$ (13)

Let $\frac{\partial\rho}{\partial\theta}_{obs}$ be the observation of $\frac{\partial\rho}{\partial\theta}$ (3) after a single episode. Using (13), we get the following:

$$\begin{aligned}
\left.\frac{\partial\rho}{\partial\theta}\right|_{obs} &= \sum_s d^\pi(s)\sum_{i=1}^M \frac{\partial\pi(s,a_i;\theta)}{\partial\theta}Q_{obs}^\pi(s,a_i) \equiv \frac{\partial\rho}{\partial\theta} + \varepsilon\\
&= \left[\sum_s d^\pi(s)\sum_{i=1}^M \frac{\partial\pi(s,a_i;\theta)}{\partial\theta}Q^\pi(s,a_i)\right] + \varepsilon\\
&= \left[\sum_s d^\pi(s)\sum_{i=1}^M \frac{\partial\pi(s,a_i;\theta)}{\partial\theta}f_{a_i}^\pi(s)\right] + \varepsilon\\
&= \left[\sum_s d^\pi(s)\sum_{i=1}^M \frac{\partial\pi(s,a_i;\theta)}{\partial\theta}\sum_{l=1}^L w_{a_i,l}\phi_{a_i,l}(s)\right] + \varepsilon\\
&= \left[\sum_{i=1}^M\sum_{l=1}^L w_{a_i,l}\left(\sum_s \frac{\partial\pi(s,a_i;\theta)}{\partial\theta}d^\pi(s)\phi_{a_i,l}(s)\right)\right] + \varepsilon \equiv \sum_{i=1}^M\sum_{l=1}^L w_{a_i,l}\Phi_{il} + \varepsilon
\end{aligned}$$

where the basis functions $\Phi_{il}$ have the form

$$\Phi_{il} = \left(\sum_s \frac{\partial\pi(s,a_i;\theta)}{\partial\theta}d^\pi(s)\phi_{a_i,l}(s)\right)$$

and $E[\varepsilon]=0$, with variance

$$V[\varepsilon] = V\left[\left.\frac{\partial\rho}{\partial\theta}\right|_{obs}\right] = \sum_s (d^\pi(s))^2\sum_{i=1}^M \left(\frac{\partial\pi(s,a_i;\theta)}{\partial\theta}\right)^2\sigma_{s,a_i}^2$$

Denoting $\widehat{\frac{\partial\rho}{\partial\theta}}_F$ as the least squares (LS) estimate of (3), its form is given by:

$$\widehat{\frac{\partial\rho}{\partial\theta}}_F = \sum_{k=1}^{ML}\beta_k\Psi_k$$ (14)

where $\beta_k$ are LS estimates of the $ML$ weights $w_{a_i,l}$ and $\Psi_k$ correspond to the $ML$ basis functions $\Phi_{il}$. Then, it can be shown that any linear system of the type given in (14) has a rate of convergence given by:

$$V\left[\widehat{\frac{\partial\rho}{\partial\theta}}_F\right] = \frac{ML}{N}V[\varepsilon] = \frac{ML}{N}\sum_s (d^\pi(s))^2\sum_{i=1}^M \left(\frac{\partial\pi(s,a_i;\theta)}{\partial\theta}\right)^2\sigma_{s,a_i}^2$$

Substituting (5) and (6) into the above equation completes the proof. □

**Proof of Theorem 2:** We prove equation (10) first. For $N$ estimates of the performance gradient, we get $N$ independent samples of each $Q_{obs}^\pi (s, a_i)$. These examples are averaged and therefore:

$$E\left[\widehat{\frac{\partial \rho}{\partial \theta}}\right] = \sum_s d^\pi (s) \sum_{i=1}^{M} \frac{\partial \pi (s, a_i; \theta)}{\partial \theta} Q^\pi (s, a_i)$$

Because each $Q_{obs}^\pi (s, a_i)$ is independently distributed, the variance of the estimate is given by

$$V\left[\widehat{\frac{\partial \rho}{\partial \theta}}\right] = \frac{1}{N} \sum_s (d^\pi (s))^2 \sum_{i=1}^{M} \left(\frac{\partial \pi (s, a_i; \theta)}{\partial \theta}\right)^2 \sigma_{s, a_i}^2 \qquad (15)$$

Given (5) the worst rate of convergence is bounded by:

$$V\left[\widehat{\frac{\partial \rho}{\partial \theta}}\right] \leq \left[\sum_s (d^\pi (s))^2 \sum_{i=1}^{M} \left(\frac{\partial \pi (s, a_i; \theta)}{\partial \theta}\right)^2\right] \sigma_{\max}^2 \frac{M}{N} = C_{\max} \frac{1}{N}$$

A similarly argument applies to the lower bound on convergence completing the proof for (10). Following the same argument for (12), we have

$$V\left[\widehat{\frac{\partial \rho}{\partial \theta}}_b\right] = \frac{1}{N} \sum_s (d^\pi (s))^2 \sum_{i=1}^{M} \left(\frac{\partial \pi (s, a_i; \theta)}{\partial \theta}\right)^2 V\left[Q^\pi (s, a_i) - \frac{1}{M} \sum_{j=1}^{M} Q^\pi (s, a_j)\right]$$

Where

$$V\left[Q^\pi (s, a_i) - \frac{1}{M} \sum_{j=1}^{M} Q^\pi (s, a_j)\right] = V\left[\frac{M-1}{M} Q^\pi (s, a_i) - \frac{1}{M} \sum_{\substack{j=1 \\ j\neq i}}^{M} Q^\pi (s, a_j)\right]$$

$$= \left(\frac{M-1}{M}\right)^2 \sigma_{s, a_i}^2 + \sum_{\substack{j=1 \\ j\neq i}}^{M} \left(\frac{1}{M}\right)^2 \sigma_{s, a_j}^2$$

$$(16)$$

Given (5) the variance $V[]$ on the far left of (16) is bounded by:

$$\left[\left(\frac{M-1}{M}\right)^2 \sigma_{s, a_i}^2 + \sum_{\substack{j=1 \\ j\neq i}}^{M} \left(\frac{1}{M}\right)^2 \sigma_{s, a_j}^2\right]_{\max} = \left(\frac{M-1}{M}\right)^2 \sigma_{\max}^2 + \sum_{\substack{j=1 \\ j\neq i}}^{M} \left(\frac{1}{M}\right)^2 \sigma_{\max}^2$$

$$= \left(\left(\frac{M-1}{M}\right)^2 + (M - 1)\left(\frac{1}{M}\right)^2\right) \sigma_{\max}^2$$

$$= \left(1 - \frac{1}{M}\right) \sigma_{\max}^2$$

Plugging the above into (16) and inserting $C_{max}$ from (6) completes the proof for the upper bound. The proof for the lower bound in the variance follows similar reasoning. $\qquad\square$